# Laplace Propagation

**Alex J. Smola, S.V.N. Vishwanathan**
Machine Learning Group
ANU and National ICT Australia
Canberra, ACT, 0200
{smola, vishy}@axiom.anu.edu.au

**Eleazar Eskin**
Department of Computer Science
Hebrew University Jerusalem
Jerusalem, Israel, 91904
eeskin@cs.columbia.edu

## Abstract

We present a novel method for approximate inference in Bayesian models and regularized risk functionals. It is based on the propagation of mean and variance derived from the Laplace approximation of conditional probabilities in factorizing distributions, much akin to Minka's Expectation Propagation. In the jointly normal case, it coincides with the latter and belief propagation, whereas in the general case, it provides an optimization strategy containing Support Vector chunking, the Bayes Committee Machine, and Gaussian Process chunking as special cases.

## 1 Introduction

Inference via Bayesian estimation can lead to optimization problems over rather large data sets. Exact computation in these cases is often computationally intractable, which has led to many approximation algorithms, such as variational approximation [5], or loopy belief propagation. However, most of these methods still rely on the propagation of the *exact* probabilities (upstream and downstream evidence in the case of belief propagation), rather than an approximation. This approach becomes costly if the random variables are real valued or if the graphical model contains large cliques.

To fill this gap, methods such as Expectation Propagation (EP) [6] have been proposed, with explicit modifications to deal with larger cliques and real-valued variables. EP works by propagating the sufficient statistics of an exponential family, that is, mean and variance for the normal distribution, between various factors of the posterior. This is an attractive choice only if we are able to compute the required quantities explicitly (this means that we need to solve an integral in closed form).

Furthermore computation of the mode of the posterior (MAP approximation) is a legitimate task in its own right — Support Vector Machines (SVM) fall into this category. In the following we develop a cheap version of EP which requires only the Laplace approximation in each step and show how this can be applied to SVM and Gaussian Processes.

**Outline of the Paper** We describe the basic ideas of LP in Section 2, show how it applies to Gaussian Processes (in particular the Bayes Committee Machine of [9]) in Section 3, prove that SVM chunking is a special case of LP in Section 4, and finally demonstrate in experiments the feasibility of LP (Section 5).

## 2  Laplace Propagation

Let $X$ be a set of observations and denote by $\theta$ a parameter we would like to infer by studying $p(\theta|X)$. This goal typically involves computing expectations $\mathbf{E}_{p(\theta|X)}[\theta]$, which can only rarely be computed exactly. Hence we approximate

$$\mathbf{E}_{p(\theta|X)}[\theta] \quad \approx \quad \operatorname{argmax}_\theta - \log p(\theta|X) =: \hat\theta \qquad (1)$$

$$\mathbf{Var}_{p(\theta|X)}[\theta] \quad \approx \quad \partial_\theta^2 \left[ - \log p(\theta|X) \right]|_{\theta=\hat\theta} \qquad (2)$$

This is commonly referred to as the Laplace-approximation. It is exact for normal distributions and works best if $\theta$ is strongly concentrated around its mean. Solving for $\hat\theta$ can be costly. However, if $p(\theta|X)$ has special structure, such as being the product of several simple terms, possibly each of them dependent only on a small number of variables at a time, computational savings can be gained. In the following we present an algorithm to take advantage of this structure by breaking up (1) into smaller pieces and optimizing over them separately.

### 2.1  Approximate Inference

For the sake of simplicity in notation we drop the explicit dependency of $\theta$ on $X$ and as in [6] we assume that

$$p(\theta) = \prod_{i=1}^{N} t_i(\theta). \qquad (3)$$

Our strategy relies on the assumption that if we succeed in finding good approximations of each of the terms $t_i(\theta)$ by $\tilde{t}_i(\theta)$ we will obtain an approximate maximizer $\tilde\theta$ of $p(\theta)$ by maximizing $\tilde{p}(\theta) := \prod_i \tilde{t}_i(\theta)$. Key is a good approximation of each of the $t_i$ at the maximum of $p(\theta)$. This is ensured by maximizing

$$\tilde{p}_i(\theta) := t_i(\theta) \prod_{j=1,j\neq i}^{N} \tilde{t}_i(\theta). \qquad (4)$$

and subsequent use of the Laplace approximation of $t_i(\theta)$ at $\tilde\theta_i := \operatorname{argmax}_\theta \tilde{p}_i(\theta)$ as the new estimate $\tilde{t}_i(\theta)$. This process is repeated until convergence. The following lemma shows that this strategy is valid:

**Lemma 1 (Fixed Point of Laplace Propagation)** *For all second-order fixed points the following holds: $\theta^*$ is a fixed point of Laplace propagation if and only if it is a local optimum of $p(\theta)$.*

**Proof**  Assume that $\theta^*$ is a fixed point of the above algorithm. Then the first order optimality conditions require $\partial_\theta \log \tilde{p}_i(\theta^*) = 0$ for all $i$ and the Laplace approximation yields $\partial_\theta \log \tilde{t}_i(\theta^*) = \partial_\theta \log t_i(\theta^*)$ and $\partial_\theta^2 \log \tilde{t}_i(\theta^*) = \partial_\theta^2 \log t_i(\theta^*)$. Consequently, up to second order, the derivatives of $\tilde{p}, \tilde{p}_i$, and $p$ agree at $\theta^*$, which implies that $\theta^*$ is a local optimum.

Next assume that $\theta^*$ is locally optimal. Then again, $\partial_\theta \log \tilde{p}_i(\theta^*)$ have to vanish, since the Laplace approximation is exact up to second order. This means that also all $\tilde{t}_i$ will have an optimum at $\tilde\theta^*$, which means that $\theta^*$ is a fixed point. ∎

The next step is to establish methods for updating the approximations $\tilde{t}_i$ of $t_i$. One option is to perform such updates sequentially, thereby improving only one $\tilde{t}_i$ at a time. This is advantageous if we can process only one approximation at a time. For parallel processing, however, we will perform several operations at a time, that is, recompute several $\tilde{t}_i(\theta)$ and merge the new approximations subsequently. We will see how the BCM is a one-step approximation of LP in the parallel case, whereas SV chunking is an exact implementation of LP in the sequential case.

## 2.2 Message Passing

Message passing [7] has been widely successful for inference in graphical models. Assume that we can split $\theta$ into a (not necessarily disjoint) set of coordinates, say $\theta_{C_1}, \ldots, \theta_{C_N}$, such that

$$p(\theta) = \prod_{i=1}^{N} t_N(\theta_{C_i}). \tag{5}$$

Then the goal of computing a Laplace approximation of $\tilde{p}_i$ reduces to computing a Laplace approximation for the subset of variables $\theta_{C_i}$, since these are the only coordinates $t_i$ depends on.

Note that an update in $\theta_{C_i}$ means that only terms sharing variables with $\theta_{C_i}$ are affected. For directed graphical models, these are the conditional probabilities governing the parents and children of $\theta_{C_i}$. Hence, to carry out calculations we only need to consider local information regarding $\tilde{t}_i(\theta_{C_i})$.

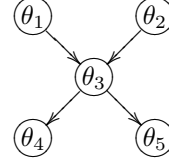

In the example above $\theta_3$ depends on $(\theta_1, \theta_2)$ and $(\theta_4, \theta_5)$ are conditionally independent of $\theta_1$ and $\theta_2$, given $\theta_3$. Consequently, we may write $p(\theta)$ as

$$p(\theta) = p(\theta_1)p(\theta_2)p(\theta_3|\theta_1,\theta_2)p(\theta_4|\theta_3)p(\theta_5|\theta_3). \tag{6}$$

To find the Laplace approximation corresponding to the terms involving $\theta_3$ we only need to consider $p(\theta_3|\theta_1,\theta_2)$ itself and its neighbors "upstream" and "downstream" of $\theta_3$ containing $\theta_1, \theta_2, \theta_3$ in their functional form.

This means that LP can be used as a drop-in replacement of exact inference in message passing algorithms. The main difference being, that now we are propagating mean and variance from the Laplace approximation rather than true probabilities (as in message passing) or true means and variances (as in expectation propagation).

# 3 Bayes Committee Machine

In this section we show that the Bayes Committee Machine (BCM) [9] corresponds to one step of LP in conjunction with a particular initialization, namely constant $\tilde{t}_i$. As a result, we extend BCM into an iterative method for improved precision of the estimates.

## 3.1 The Basic Idea

Let us assume that we are given a set of sets of observations, say, $Z_1, \ldots, Z_N$, which are conditionally independent of each other, given a parameter $\theta$, as depicted in the figure on the right.

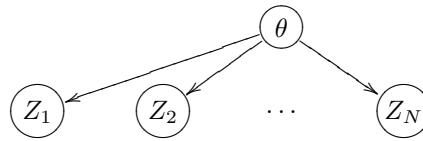

Repeated application of Bayes rule allows us to rewrite the conditional density $p(\theta|Z)$ as

$$p(\theta|Z) \propto p(Z|\theta)p(\theta) = p(\theta) \prod_{i=1}^{N} p(Z_i|\theta) \propto p^{1-N}(\theta) \prod_{i=1}^{N} p(\theta|Z_i). \tag{7}$$

Finally, Tresp and coworkers [9] find Laplace approximations for $p(\theta|Z_i) \propto p(Z_i|\theta)p(\theta)$ with respect to $\theta$. These results are then combined via (7) to come up with an overall estimate of $p(\theta|X, Y)$.

## 3.2 Rewriting The BCM

The repeated invocation of Bayes rule seems wasteful, yet it was necessary in the context of the BCM formulation to explain how estimates from subsets could be combined in a committee like fashion. To show the equivalence of BCM with one step of LP recall the third term of (7). We have

$$p(\theta|Z) = \underbrace{c \cdot p(\theta)}_{:=t_0(\theta)} \prod_{i=1}^{N} \underbrace{p(Z_i|\theta)}_{:=t_i(\theta)}, \tag{8}$$

where $c$ is a suitable normalizing constant. In Gaussian processes, we generally assume that $p(\theta)$ is normal, hence $t_0(\theta)$ is quadratic. This allows us to state the LP algorithm to find the mode and curvature of $p(\theta|Z)$:

---

**Algorithm 1** Iterated Bayes Committee Machine

---

Initialize $\tilde{t}_0 \leftarrow cp(\theta)$ and $\tilde{t}_i(\theta) \leftarrow \text{const.}$

**repeat**

Compute new approximations $\tilde{t}_i(\theta)$ in parallel by finding Laplace approximations to $\tilde{p}_i$, as defined in (4). Since $t_0$ is normal, $\tilde{t}_0(\theta) = t_0(\theta)$. For $i \neq 0$ we obtain

$$\tilde{p}_i = t_i(\theta) \prod_{j=0, j \neq i}^{N} \tilde{t}_i(\theta) = p(\theta)p(Z_i|\theta) \prod_{j=1, j \neq i}^{N} \tilde{t}_i(\theta). \tag{9}$$

**until** Convergence

Return $\text{argmax}_\theta \, t_0(\theta) \prod_{i=1}^{N} \tilde{t}_i(\theta)$.

---

Note that in the first iteration (9) can be written as $\tilde{p}_i \propto p(\theta)p(Z_i|\theta)$, since all remaining terms $\tilde{t}_i$ are constant. This means that after the first update $\tilde{t}_i$ is identical to the estimates obtained from the BCM.

Whereas the BCM stops at this point, we have the liberty to continue the approximation and also the liberty to choose whether we use a parallel or a sequential update regime, depending on the number of processing units available. As a side-effect, we obtain a simplified proof of the following:

**Theorem 2 (Exact BCM [9])** *For normal distributions the BCM is exact, that is, the Iterated BCM converges in one step.*

**Proof** For normal distributions all $\tilde{t}_i$ are exact, hence $p(\theta) = \prod_i t_i(\theta) = \prod_i \tilde{t}_i(\theta) = \tilde{p}(\theta)$, which shows that $\tilde{p} = p$. ∎

Note that [9] formulates the problem as one of classification or regression, that is $Z = (X, Y)$, where the labels $Y$ are conditionally independent, given $X$ and the parameter $\theta$. This, however, does not affect the validity of our reasoning.

## 4 Support Vector Machines

The optimization goals in Support Vector Machines (SVM) are very similar to those in Gaussian Processes: essentially the negative log posterior $-\log p(\theta|Z)$ corresponds to the objective function of the SV optimization problem.

This gives hope that LP can be adapted to SVM. In the following we show that SVM chunking [4] and parallel SVM training [2] can be found to be special cases of LP. Taking logarithms of (3) and defining $\pi_i(\theta) := -\log t_i(\theta)$ (and $\tilde{\pi}(\theta) := -\log \tilde{t}_i(\theta)$ analogously) we obtain the following formulation of LP in log-space.

---

**Algorithm 2** Logarithmic Version of Laplace Propagation

---
Initialize $\tilde{\pi}_i(\theta)$
**repeat**
    Choose index $i \in \{1, \ldots, N\}$
    Minimize $\pi_i(\theta) + \sum_{j=1, i\neq j}^{N} \tilde{\pi}_j(\theta)$ and replace $\tilde{\pi}_i(\theta)$ by a Taylor approximation at the
    minimum $\theta_i$ of the above expression.
**until** All $\theta_i$ agree

---

### 4.1 Chunking

To show that SV chunking is equivalent to LP in logspace, we briefly review the basic ideas of chunking. The standard SVM optimization problem is

$$\begin{aligned} \underset{\theta, b}{\text{minimize}} \quad & \pi(\theta, b) := \frac{1}{2}\|\theta\|^2 + C\sum_{i=1}^{m} c(x_i, y_i, f(x_i)) \\ \text{subject to} \quad & f(x_i) = \langle \theta, \Phi(x_i)\rangle + b \end{aligned} \tag{10}$$

Here $\Phi(x)$ is the map into feature space such that $k(x, x') = \langle \Phi(x), \Phi(x')\rangle$ and $c(x, y, f(x))$ is a loss function penalizing the deviation between the estimate $f(x)$ and the observation $y$. We typically assume that $c$ is convex. For the rest of the deviation we let $c(x, y, f(x)) = \max(0, 1 - yf(x))$ (the analysis still holds in the general case, however it becomes considerably more tedious). The dual of (10) becomes

$$\underset{\alpha}{\text{minimize}} \ \frac{1}{2}\sum_{i,j=1}^{m} \alpha_i \alpha_j y_i y_j K_{ij} k(x_i, x_j) - \sum_{i=1}^{m} \alpha_i \ \text{s.t.} \ \sum_{i=1}^{m} y_i\alpha_i = 0 \text{ and } \alpha_i \in [0, C] \tag{11}$$

The basic idea of chunking is to optimize only over subsets of the vector $\alpha$ at a time.

Denote by $S_w$ the set of variables we are using in the current optimization step, let $\alpha_w$ be the corresponding vector, and by $\alpha_f$ the variables which remain unchanged. Likewise denote by $y_w, y_f$ the corresponding parts of $y$, and let $H = \begin{bmatrix} H_{ww} & H_{wf} \\ H_{fw} & H_{ff} \end{bmatrix}$ be the quadratic matrix of (11), again split into terms depending on $\alpha_w$ and $\alpha_f$ respectively. Then (11), restricted to $\alpha_w$ can be written as [4]

$$\underset{\alpha_w}{\text{minimize}} \ \frac{1}{2}\alpha_w^\top H_{ww}\alpha_w + \alpha_f^\top H_{fw}\alpha_w - \sum_{i\in S_w} \alpha_i \ \text{s.t.} \ y_w^\top \alpha_w + y_f^\top \alpha_f = 0, \ \alpha_i \in [0, C] \tag{12}$$

### 4.2 Equivalence to LP

We now show that the correction terms arising from chunking are the same as those arising from LP. Denote by $S_1, \ldots, S_N$ a partition of $\{1, \ldots m\}$ and define

$$\pi_0(\theta, b) := \frac{1}{2}\|\theta\|^2 \text{ and } \pi_i(\theta, b) := C\sum_{j\in S_i} c(x_j, y_j, f(x_j)). \tag{13}$$

Then $\tilde{\pi}_0 = \pi_0$, since $\pi_0$ is purely quadratic, regardless of where we expand $\pi_0$. As for $\pi_i$ (with $i \neq 0$) we have

$$\tilde{\pi}_i = \sum_{j\in S_i} y_j \beta_j \langle \Phi(x_j), \theta\rangle + \sum_{j\in S_i} y_j \beta_j b = \langle \theta_i, \theta\rangle + b_i b \tag{14}$$

where $\beta_j \in Cc'(x_j, y_j, f(x_j))$, $\theta_i := \sum_{j \in S_i} y_j \beta_j \Phi(x_j)$, and $b_i := \sum_{j \in S_i} y_j \beta_j$.[1] In this case minimization over $\pi_i(\theta) + \sum_{j \neq i} \tilde{\pi}_j(\theta)$ amounts to minimizing

$$\frac{1}{2}\|\theta\|^2 + C \sum_{j \in S_i} c(x_j, y_j, f(x_j)) + C \sum_{j \notin S_i} [\langle \theta_j, \theta \rangle + b_j b] \text{ s.t. } f(x_j) = \langle \theta, \Phi(x_j) \rangle + b.$$

Skipping technical details, the dual optimization problem is given by

$$
\begin{aligned}
\underset{\alpha}{\text{minimize}} \quad & \frac{1}{2} \sum_{j,l \in S_i} \alpha_j \alpha_l y_j y_l k(x_j, k_l) - \sum_{j \in S_i} \alpha_j - \sum_{j \in S_i, l \notin S_i} \alpha_j \beta_l y_j y_l k(x_j, k_l) \\
\text{subject to} \quad & \alpha_j \in [0, C] \text{ and } \sum_{j \in S_i} y_j \alpha_j - \sum_{j \notin S_i} y_j \beta_j = 0.
\end{aligned}
\tag{15}
$$

The latter is identical to (12), the optimization problem arising from chunking, provided that we perform the substitution $\alpha_j = -\beta_j$ for all $j \notin S_i$.

To show this last step, note that at optimality null has to be an element of the subdifferential of $\pi_i(\theta)$ with respect to $\theta, b$. Taking derivatives of $\pi_i + \sum_{j \neq i} \tilde{\pi}_i$ implies

$$\theta \in -C \sum_{j \in S_i} c'(x_j, y_j, f(x_j)) - C \sum_{j \neq i} \theta_j. \tag{16}$$

Matching up terms in the expansion of $\theta$ we immediately obtain $\beta_j = -\alpha_j$.

Finally, to start the approximation scheme we need to consider a proper initialization of $\tilde{\pi}_i$. In analogy to the BCM setting we use $\tilde{\pi}_i = 0$, which leads precisely to the SVM chunking method, where one optimizes over one subset at a time (denoted by $S_i$), while the other sets are fixed, taking only their linear contribution into account.

LP does not require that all the updates of $\tilde{t}_i$ (or $\tilde{\pi}_i$) be carried out sequentially. Instead, we can also consider parallel approximations similar to [2]. There the optimization problem is split into several small parts and each of them is solved independently. Subsequently the estimates are combined by averaging.

This is equivalent to one-step parallel LP: with the initialization $\tilde{\pi}_i = 0$ for all $i \neq 0$ and $\tilde{\pi}_0 = \pi_0 = \frac{1}{2}\|\theta\|^2$ we minimize $\pi_i + \sum_{j \neq i} \tilde{\pi}_j$ in parallel. This is equivalent to solving the SV optimization problem on the corresponding subset $S_i$ (as we saw in the previous section). Hence, the linear terms $\theta_i, b_i$ arising from the approximation $\tilde{\pi}_i(\theta, b) = C\langle \theta_i, \theta \rangle + Cb_i b$ lead to the overall approximation

$$\tilde{\pi}(\theta, b) = \sum_i \tilde{\pi}_i(\theta, b) = \frac{1}{2}\|\theta\|^2 + \sum_i \langle \theta_i, \theta \rangle, \tag{17}$$

with the joint minimizer being the average of the individual solutions.

## 5 Experiments

To test our ideas we performed a set of experiments with the widely available Web and Adult datasets from the UCI repository [1]. All experiments were performed on a 2.4 MHz Intel Xeon machine with 1 GB RAM using MATLAB R13. We used a RBF kernel with $\sigma^2 = 10$ [8], to obtain comparable results.

We first tested the performance of Gaussian process training with Laplace propogation using a logistic loss function. The data was partitioned into chunks of roughly 500 samples each and the maximum of columns in the low rank approximation [3] was set to 750.

We summarize the performance of our algorithm in Table 1. $T_{\text{Factor}}$ refers to the time (in seconds) for computing the low rank factorization while $T_{\text{Train}}$ denotes the training time for the Gaussian process. We empirically observed that on all datasets the algorithm converges in less than 3 iterations using serial updates and in less than 6 iterations using parallel updates.

| Dataset | $T_{\text{Factor}}$ | $T_{\text{Serial}}$ | $T_{\text{Parallel}}$ | Dataset | $T_{\text{Factor}}$ | $T_{\text{Serial}}$ | $T_{\text{Parallel}}$ |
|---------|------|--------|----------|------|--------|--------|---------|
| Adult1 | 16.38 | 25.72 | 53.90 | Web1 | 20.33 | 34.33 | 93.47 |
| Adult2 | 20.07 | 33.02 | 75.76 | Web2 | 36.27 | 67.65 | 88.37 |
| Adult3 | 24.41 | 47.05 | 106.88 | Web3 | 37.09 | 92.36 | 212.04 |
| Adult4 | 36.29 | 75.71 | 202.88 | Web4 | 69.9 | 168.88 | 251.92 |
| Adult5 | 56.82 | 97.57 | 169.79 | Web5 | 68.15 | 225.13 | 249.15 |
| Adult6 | 89.78 | 232.45 | 348.10 | Web6 | 129.86 | 261.23 | 663.07 |
| Adult7 | 119.39 | 293.45 | 559.23 | Web7 | 213.54 | 483.52 | 838.36 |

Table 1: Gaussian process training with serial and parallel Laplace propogation.

We conducted another set of experiments to test the speedups obtained by *seeding* the SMO with values of $\alpha$ obtained by performing one iteration of Laplace propogation on the dataset. As before we used a RBF kernel with $\sigma^2 = 10$. We partitioned the Adult1 and Web1 datasets into 5 chunks each while the Adult4 and Web4 datasets were partitioned into 10 chunks each. The freely available SMOBR package was modified and used for our experiments. For simplicity we use the C-SVM and vary the regularization parameter. $T_{\text{Parallel}}$, $T_{\text{Serial}}$ and $T_{\text{NoMod}}$ refer to the times required by SMO to converge when using one iteration of parallel/serial/no LP on the dataset.

| | Adult1 | | | | Adult4 | | |
|---|--------|---------|---------|---|--------|---------|---------|
| C | $T_{\text{Parallel}}$ | $T_{\text{Serial}}$ | $T_{\text{NoMod}}$ | C | $T_{\text{Parallel}}$ | $T_{\text{Serial}}$ | $T_{\text{NoMod}}$ |
| 0.1 | 2.84 | 2.04 | 7.650 | 0.1 | 20.42 | 13.26 | 59.935 |
| 0.5 | 5.57 | 3.99 | 9.215 | 0.5 | 46.29 | 40.82 | 63.645 |
| 1.0 | 5.48 | 7.25 | 10.885 | 1.0 | 80.33 | 64.37 | 107.475 |
| 5.0 | 107.37 | 110.07 | 307.135 | 5.0 | 1921.19 | 1500.42 | 1427.925 |

Table 2: Performance of SMO Initialization on the Adult dataset.

| | Web1 | | | | Web4 | | |
|---|--------|---------|---------|---|--------|---------|---------|
| C | $T_{\text{Parallel}}$ | $T_{\text{Serial}}$ | $T_{\text{NoMod}}$ | C | $T_{\text{Parallel}}$ | $T_{\text{Serial}}$ | $T_{\text{NoMod}}$ |
| 0.1 | 21.36 | 15.65 | 27.34 | 0.1 | 63.76 | 77.05 | 95.10 |
| 0.5 | 34.64 | 35.66 | 60.12 | 0.5 | 140.61 | 149.80 | 156.525 |
| 1.0 | 61.15 | 38.56 | 63.745 | 1.0 | 254.84 | 298.59 | 232.120 |
| 5.0 | 224.15 | 62.41 | 519.67 | 5.0 | 1959.08 | 3188.75 | 2223.225 |

Table 3: Performance of SMO Initialization on the Web dataset.

As can be seen our initialization significantly speeds up the SMO in many cases sometimes acheiving upto 4 times speed up. Although in some cases (esp for large values of $C$) our method seems to slow down convergence of SMO. In general serial updates seem to perform better than parallel updates. This is to be expected since we use the information from other blocks as soon as they become available in case of the serial algorithm while we completely ignore the other blocks in the parallel algorithm.

# 6   Summary And Discussion

Laplace propagation fills the gap between Expectation Propagation, which requires exact computation of first and second order moments, and message passing algorithms when optimizing structured density functions. Its main advantage is that it only requires the Laplace approximation in each computational step, while being applicable to a wide range of optimization tasks. In this sense, it complements Minka's Expectation Propagation, whenever exact expressions are not available.

As a side effect, we showed that Tresp's Bayes Committee Machine and Support Vector Chunking methods are special instances of this strategy, which also sheds light on the fact why simple averaging schemes for SVM, such as the one of Colobert and Bengio seem to work in practice.

The key point in our proofs was that we split the data into disjoint subsets. By the assumption of independent and identically distributed data it followed that the variable assignments are conditionally independent from each other, given the parameter $\theta$, which led to a favorable factorization property in $p(\theta|Z)$. It should be noted that LP allows one to perform chunking-style optimization in Gaussian Processes, which effectively puts an upper bound on the amount of memory required for optimization purposes.

**Acknowledgements**   We thank Nir Friedman, Zoubin Ghahramani and Adam Kowalczyk for useful suggestions and discussions.

## Footnotes

[1] Note that we had to replace the equality with set inclusion due to the fact that $c$ is not everywhere differentiable, hence we used sub-differentials instead.

# References

[1] C. L. Blake and C. J. Merz. UCI repository of machine learning databases, 1998.

[2] R. Collobert, S. Bengio, and Y. Bengio. A parallel mixture of svms for very large scale problems. In *Advances in Neural Information Processing Systems*. MIT Press, 2002.

[3] S. Fine and K. Scheinberg. Efficient SVM training using low-rank kernel representations. *Journal of Machine Learning Research*, 2:243–264, Dec 2001. http://www.jmlr.org.

[4] T. Joachims. Making large-scale SVM learning practical. In B. Schölkopf, C. J. C. Burges, and A. J. Smola, editors, *Advances in Kernel Methods—Support Vector Learning*, pages 169–184, Cambridge, MA, 1999. MIT Press.

[5] M. I. Jordan, Z. Gharamani, T. S. Jaakkola, and L. K. Saul. An introduction to variational methods for graphical models. In *Learning in Graphical Models*, volume M. I. Jordan, pages 105–162. Kluwer Academic, 1998.

[6] T. Minka. *Expectation Propagation for approximative Bayesian inference*. PhD thesis, MIT Media Labs, Cambridge, USA, 2001.

[7] J. Pearl. *Probabilistic Reasoning in Intelligent Systems*. Morgan-Kaufman, 1988.

[8] J. C. Platt. Sequential minimal optimization: A fast algorithm for training support vector machines. Technical Report MSR-TR-98-14, Microsoft Research, 1998.

[9] V. Tresp. A Bayesian committee machine. *Neural Computation*, 12(11):2719–2741, 2000.